# Event-Driven Simulation of Networks of Spiking Neurons

Lloyd Watts
Synaptics Inc.
2698 Orchard Parkway
San Jose CA 95134
lloyd@synaptics.com

## Abstract

A fast event-driven software simulator has been developed for simulating large networks of spiking neurons and synapses. The primitive network elements are designed to exhibit biologically realistic behaviors, such as spiking, refractoriness, adaptation, axonal delays, summation of post-synaptic current pulses, and tonic current inputs. The efficient event-driven representation allows large networks to be simulated in a fraction of the time that would be required for a full compartmental-model simulation. Corresponding analog CMOS VLSI circuit primitives have been designed and characterized, so that large-scale circuits may be simulated prior to fabrication.

## 1   Introduction

Artificial neural networks typically use an abstraction of real neuron behaviour, in which the continuously varying mean firing rate of the neuron is presumed to carry the information about the neuron's time-varying state of excitation [1]. This useful simplification allows the neuron's state to be represented as a time-varying continuous-amplitude quantity. However, spike timing is known to be important in many biological systems. For example, in nearly all vertebrate auditory systems, spiral ganglion cells from the cochlea are known to phase lock to pure-tone stimuli for all but the highest perceptible frequencies [2]. The barn owl uses axonal delays to compute azimuthal spatial localization [3]. Studies in the cat [4] have shown that

relative timing of spikes is preserved even at the highest cortical levels. Studies in the visual system of the blowfly [5] have shown that the information contained in just three spikes is enough for the fly to make a decision to turn, if the visual input is sparse.

Thus, it is apparent that biological neural systems exploit the spiking and time-dependent behavior of the neurons and synapses to perform system-level computation. To investigate this type of computation, we need a simulator that includes detailed neural behavior, yet uses a signal representation efficient enough to allow simulation of large networks in a reasonable time.

## 2    Spike: Event-Driven Simulation

Spike is a fast event-driven simulator optimized for simulating networks of spiking neurons and synapses. The key simplifying assumption in Spike is that all currents injected into a cell are composed of piecewise-constant pulses (i.e., boxcar pulses), and therefore all integrated membrane voltage trajectories are piecewise linear in time. This very simple representation is capable of surprisingly complex and realistic behaviors, and is well suited for investigating system-level questions that rely on detailed spiking behavior.

The simulator operates by maintaining a queue of scheduled events. The occurrence of one event (i.e., a neuron spike) usually causes later events to be scheduled in the queue (i.e., end of refractory period, end of post-synaptic current pulse). The total current injected into a cell is integrated into the future to predict the time of firing, at which time a neuron spike event is scheduled. If any of the current components being injected into the cell subsequently change, the spike event is rescheduled. The simulator runs until the queue is empty or until the desired run-time has elapsed. A similar event-driven neural simulator was developed by Pratt [6].

The simulator output may be plotted by a number of commercially available plotting programs, including Gnuplot, Mathematica, Xvgr, and Cview.

## 3    NeuraLOG: Neural Schematic Capture

NeuraLOG is a schematic entry tool, which allows the convenient entry of "neural" circuit diagrams, consisting of neurons, synapses, test inputs, and custom symbols. NeuraLOG is a customization of the program AnaLOG, by John Lazzaro and Dave Gillespie.

The parameters of the neural elements are entered directly on the schematic diagram; these parameters include the neuron refractory period, duration and intensity of the post-synaptic current pulse following an action potential, saturation value of summating post-synaptic currents, tonic input currents, axonal delays, etc. Custom symbols can be defined, so that arbitrarily complex hierarchical designs may be made. It is common to create a complex "neuron" containing many neuron and synapse primitive elements. Spiking inputs may be supplied as external stimuli for the circuit in a number of different formats, including single spikes, periodic spike trains, periodic bursts, poisson random spike trains, and gaussian-jittered periodic

spike trains. Textual input to Spike is also supported, to allow simulation of circuit topologies that would be too time-consuming to enter graphically.

## 4   A Simple Example

A simple example of a neural circuit is shown in Figure 1. This circuit consists of two neurons (the large disks), several synapses (the large triangles), and two tonic inputs (the small arrows). The text strings associated with each symbol define that symbol's parameters: neuron parameters are identifier labels (i.e., **n1**) and refractory period in milliseconds (ms); synapse parameters are the value of the post-synaptic current in nA, and the duration of the current pulse in ms, and an optional saturation parameter, which indicates how many post-synaptic current pulses may be superposed before saturation; the tonic input parameter is the injected current in nA. Filled symbols (tonic inputs and synapses) indicate inhibitory behavior.

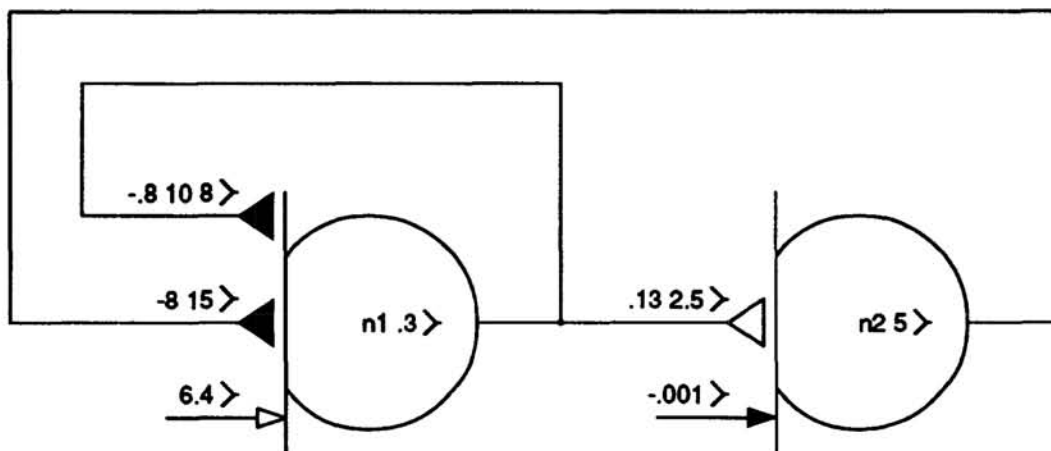

Figure 1: Graphical input representation of a simple neural circuit, as entered in NeuraLOG.

The simulated behavior of the circuit is shown in Figure 2. The neuron **n1** exhibits an adapting bursting behavior, as seen in the top trace of the plot.

The excitatory tonic current input to neuron **n1** causes **n1** to fire repeatedly. The weakly excitatory synapse from **n1** to neuron **n2** causes **n2** to fire after many spikes from **n1**. The synaptic current in the synapse from **n1** to **n2** is plotted in the trace labeled **sn1n2**. The strongly inhibitory synapse from **n2** to **n1** causes **n1** to stop firing after **n2** fires a spike. The synaptic current in the synapse from **n2** to **n1** is plotted in the trace labeled **sn2n1**. The combination of the excitatory tonic input to **n1** and the inhibitory feedback from **n2** to **n1** causes the bursting behavior.

The adapting behavior is caused by the self-inhibitory accumulating feedback from neuron **n1** to itself, via the summating inhibitory synapse in the top left of the diagram. Each spike on **n1** causes a slightly increased inhibitory current into **n1**, which gradually slows the rate of firing with each successive pulse. The synaptic current in this inhibitory synapse is plotted in the trace **sn1n1**; it is similar to the

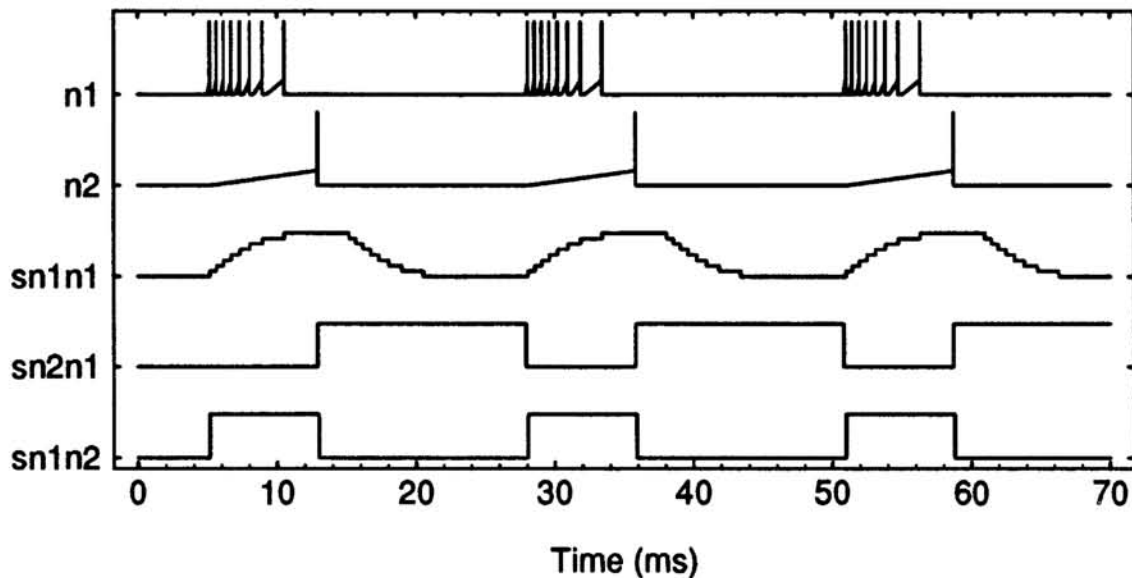

Figure 2: Simulation results for the circuit of Figure 1, showing adapting bursting behavior.

current that would be generated by a calcium-dependent potassium channel.

This simple example demonstrates that the summating synapse primitive can be used to model a behavior that is not strictly synaptic in origin; it can be thought of as a general time-dependent state variable. This example also illustrates the principle that proper network topology (summating synapse in a negative feedback loop) can lead to realistic system-level behavior (gradual adaptation), even though the basic circuit elements may be rather primitive (boxcar current pulses).

## 5    Applications of the Simulation Tools

NeuraLOG and Spike have been used by the author to model spiking associative memories, adaptive structures that learn to predict a time delay, and chaotic spiking circuits. Researchers at Caltech [7, 8] and the Salk Institute have used the tools in their studies of locust central pattern generators (CPGs) and cortical oscillations. The cortical oscillation circuits contain a few hundred neurons and a few thousand synapses. A CPG circuit, developed by Sylvie Ryckebusch, is shown in Figure 3; the corresponding simulation output is shown in Figure 4.

NeuraLOG and Spike are distributed at no charge under the GNU licence. They are currently supported on HP and Sun workstations. The tools are supplied with a user's manual and working tutorial examples.

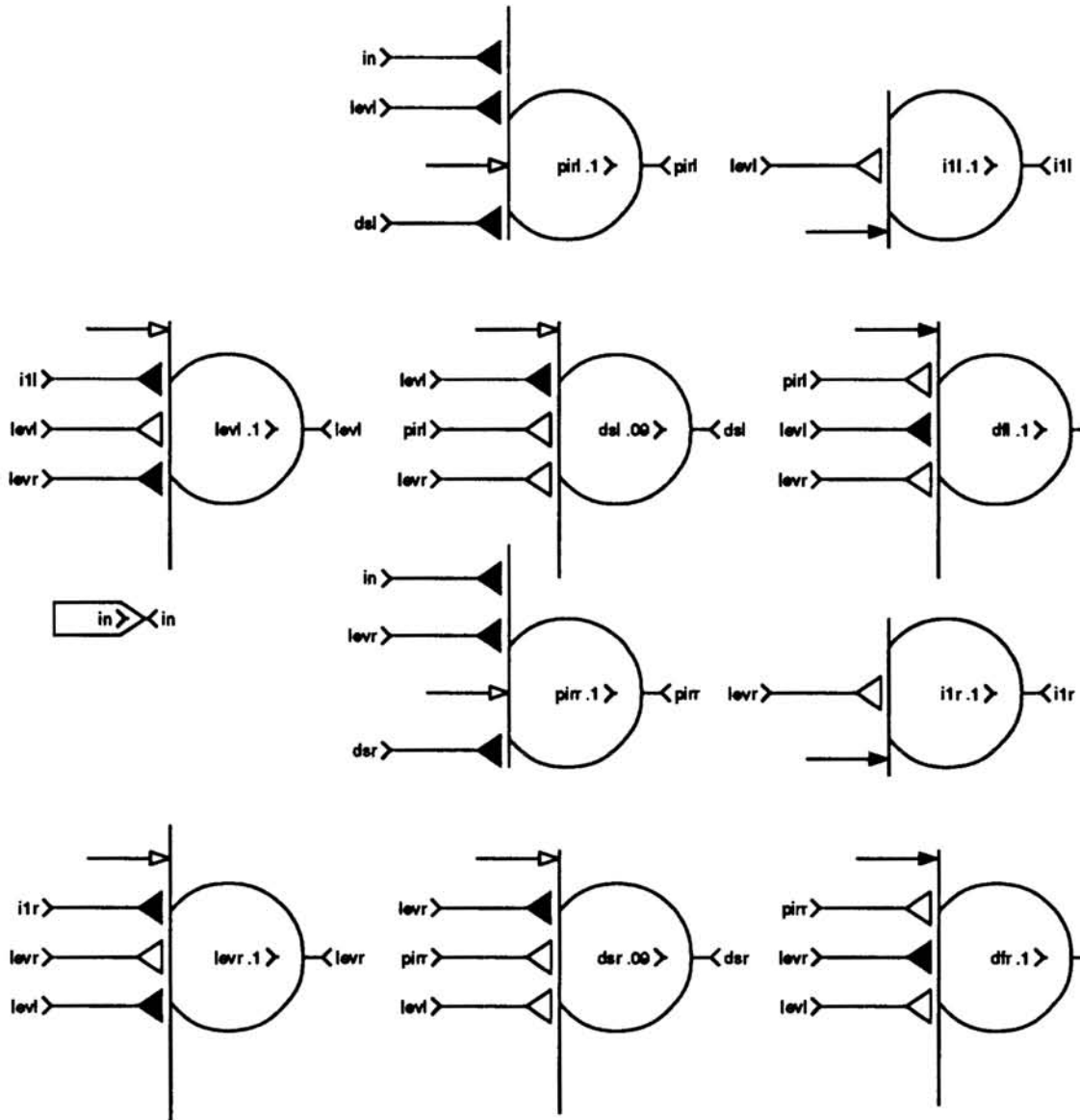

Figure 3: Sylvie Ryckebusch's locust CPG circuit. For clarity, the synapse parameters have been omitted.

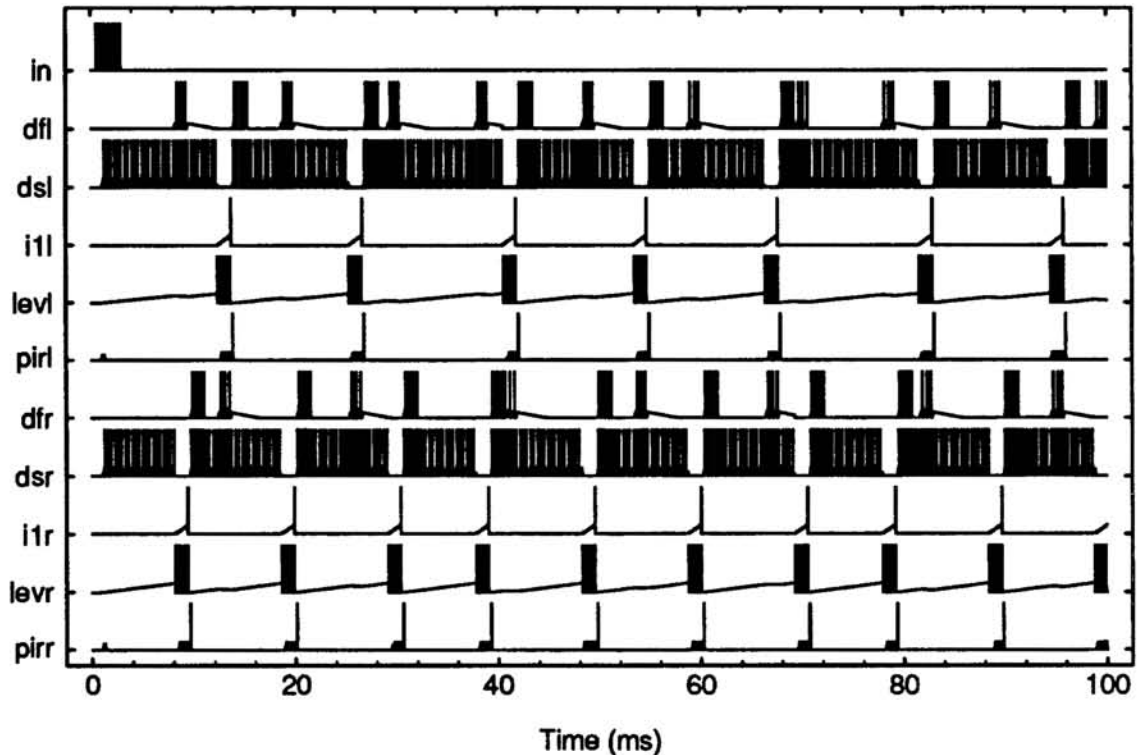

**Time (ms)**

Figure 4: Simulation results for Sylvie Ryckebusch's locust CPG circuit.

# 6   The Link to Analog VLSI

Analog VLSI circuit primitives that can be modelled by Spike have been designed and tested. The circuits are shown in Figure 5, and have been described previously [9, 10]. These circuits have been used by workers at Caltech to implement VLSI models of central pattern generators. The software simulation tools allow simulation of complex neural circuits prior to fabrication, to improve the likelihood of success on first silicon, and to allow optimization of shared parameters (bias wires).

# 7   Conclusion

NeuraLOG and Spike fill a need for a fast neural simulator that can model large networks of biologically realistic spiking neurons. The simple computational primitives within Spike can be used to create complex and realistic neural behaviors in arbitrarily complex hierarchical designs. The tools are publicly available at no charge. NeuraLOG and Spike have been used by a number of research labs for detailed modeling of biological systems.

**Acknowledgements**

NeuraLOG is a customization of the program AnaLOG, which was written by John Lazzaro and David Gillespie. Lloyd Watts gratefully acknowledges helpful discussions with Carver Mead, Sylvie Ryckebusch, Misha Mahowald, John Lazzaro,

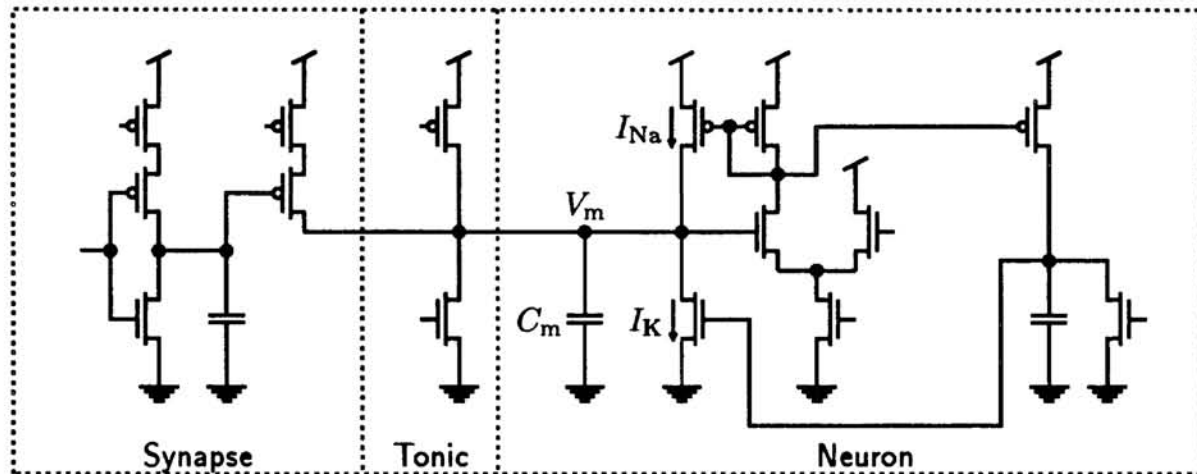

Figure 5: CMOS Analog VLSI circuit primitives. The neuron circuit models a voltage-gated sodium channel and a delayed rectifier potassium channel to produce a spiking mechanism. The tonic circuit allows constant currents to be injected onto the membrane capacitance $C_m$. The synapse circuit creates a boxcar current pulse in response to a spike input.

David Gillespie, Mike Vanier, Brad Minch, Rahul Sarpeshkar, Kwabena Boahen, John Platt, and Steve Nowlan. Thanks to Sylvie Ryckebusch for permission to use her CPG circuit example.

# References

[1] J. Hertz, A. Krogh and R. Palmer, *Introduction to the Theory of Neural Computation*, Addison-Wesley, 1991.

[2] N. Y-S. Kiang, T. Watanabe, E. C. Thomas, L. F. Clark, "Discharge Patterns of Single Fibers in the Cat's Auditory Nerve", MIT Res. Monograph No. 35, (MIT, Cambridge, MA).

[3] M. Konishi, T.T. Takahashi, H. Wagner, W.E. Sullivan, C.E. Carr, "Neurophysiological and Anatomical Substrates of Sound Localization in the Owl", In *Auditory Function*, G.M. Edelman, W.E. Gall, and W.M. Cowan, eds., pp. 721–745, Wiley, New York.

[4] D. P. Phillips and S. E. Hall, "Response Timing Constraints on the Cortical Representation of Sound Time Structure", Journal of the Acoustical Society of America, **88** (3), pp. 1403–1411, 1990.

[5] R.R. de Ruyter van Steveninck and W. Bialek, "Real-time Performance of a movement-sensitive neuron in the blowfly visual system: Coding and infor-

mation transfer in short spike sequences", *Proceedings of the Royal Society of London, Series B,* **234**, 379-414.

[6] G. A. Pratt, *Pulse Computation*, Ph.D. Thesis, Massachusetts Institute of Technology, 1989.

[7] M. Wehr, S. Ryckebusch and G. Laurent, Western Nerve Net Conference, Seattle, Washington, 1993.

[8] S. Ryckebusch, M. Wehr, and G. Laurent, "Distinct rhythmic locomotor patterns can be generated by a simple adaptive neural circuit: biology, simulation, and VLSI implementation", in review, Journal of Computational Neuroscience.

[9] R. Sarpeshkar, L. Watts, C.A. Mead, "Refractory Neuron Circuits", Internal Memorandum, Physics of Computation Laboratory, California Institute of Technology, 1992.

[10] L. Watts, "Designing Networks of Spiking Silicon Neurons and Synapses", Proceedings of Computation and Neural Systems Meeting CNS*92, San Francisco, CA, 1992.
